# Plasticity of Center-Surround Opponent Receptive Fields in Real and Artificial Neural Systems of Vision

**S. Yasui**
Kyushu Institute of Technology
Iizuka 820, Japan

**T. Furukawa**
Kyushu Institute of Technology
Iizuka 820, Japan

**M. Yamada**
Electrotechnical Laboratory
Tsukuba 305, Japan

**T. Saito**
Tsukuba University
Tsukuba 305, Japan

## Abstract

Despite the phylogenic and structural differences, the visual systems of different species, whether vertebrate or invertebrate, share certain functional properties. The center-surround opponent receptive field (CSRF) mechanism represents one such example. Here, analogous CSRFs are shown to be formed in an artificial neural network which learns to localize contours (edges) of the luminance difference. Furthermore, when the input pattern is corrupted by a background noise, the CSRFs of the hidden units becomes shallower and broader with decrease of the signal-to-noise ratio (SNR). The same kind of SNR-dependent plasticity is present in the CSRF of real visual neurons; in bipolar cells of the carp retina as is shown here experimentally, as well as in large monopolar cells of the fly compound eye as was described by others. Also, analogous SNR-dependent plasticity is shown to be present in the biphasic flash responses (BPFR) of these artificial and biological visual systems. Thus, the spatial (CSRF) and temporal (BPFR) filtering properties with which a wide variety of creatures see the world appear to be optimized for detectability of changes in space and time.

## 1  INTRODUCTION

A number of learning algorithms have been developed to make synthetic neural machines be trainable to function in certain optimal ways. If the brain and nervous systems that we see in nature are best answers of the evolutionary process, then one might be able to find some common 'softwares' in real and artificial neural systems. This possibility is examined in this paper, with respect to a basic visual

mechanism relevant to detection of brightness contours (edges). In most visual systems of vertebrate and invertebrate, one finds interneurons which possess center-surround opponent receptive fields (CSRFs). CSRFs underlie the mechanism of lateral inhibition which produces edge enhancement effects such as Mach band. It has also been shown in the fly compound eye that the CSRF of large monopolar cells (LMCs) changes its shape in accordance with SNR; the CSRF becomes wider with increase of the noise level in the sensory environment. Furthermore, whereas CSRFs describe a filtering function in space, an analogous observation has been made in LMCs as regards the filtering property in the time domain; the biphasic flash response (BPFR) lasts longer as the noise level increases (Dubs, 1982; Laughlin, 1982).

A question that arises is whether similar SNR-dependent spatio-temporal filtering properties might be present in vertebrate visual cells. To investigate this, we made an intracellular recording experiment to measure the CSRF and BPFR profiles of bipolar cells in the carp retina under appropriate conditions, and the results are described in the first part of this paper. In the second part, we ask the same question in a 3-layer feedforward artificial neural network (ANN) trained to detect and localize spatial and temporal changes in simulated visual inputs corrupted by noise. In this case, the ANN wiring structure evolves from an initial random state so as to minimize the detection error, and we look into the internal ANN organization that emerges as a result of training. The findings made in the real and artificial neural systems are compared and discussed in the final section.

In this study, the backpropagation learning algorithm was applied to update the synaptic parameters of the ANN. This algorithm was used as a means for the computational optimization. Accordingly, the present choice is not necessarily relevant to the question of whether the error backpropagation pathway actually might exist in real neural systems(cf. Stork & Hall, 1989).

## 2   THE CASE OF A REAL NEURAL SYSTEM: RETINAL BIPOLAR CELL

Bipolar cells occur as a second order neuron in the vertebrate retina, and they have a good example of CSRF Here we are interested in the possibility that the CSRF and BPFR of bipolar cells might change their size and shape as a function of the visual environment, particularly as regards the dark- versus light-adapted retinal states which correspond to low versus high SNR conditions as explained later. Thus, the following intracellular recording experiment was carried out.

### 2.1   MATERIAL AND METHOD

The retina was isolated from the carp which had been kept in complete darkness for 2 hrs before being pithed for sacrifice. The specimen was then mounted on a chamber with the receptor side up, and it was continuously superfused with a Ringer solution composed of (in mM) 102 NaCl, 28 $NaHCO_3$, 2.6 KCl, 1 $CaCl_2$, 1 $MgCl_2$ and 5 glucose, maintained at pH=7.6 and aerated with a gas mixture of 95% $O_2$ and 5% $CO_2$. Glass micropipettes filled with 3M KCl and having tip resistances of about 150 $M\Omega$ were used to record the membrane potential. Identification of bipolar cell units was made on the basis of presence or absence of CSRF. For this preliminary test, the center and peripheral responses were examined by using flashes of a small centered spot and a narrow annular ring. To map their receptive field profile, the stimulus was given as flashes of a narrow slit presented at discrete positions 60 $\mu$m apart on the retina. The slit of white light was 4 mm long and 0.17 mm wide, and its flash had intensity of 7.24 $\mu W/cm^2$ and duration of 250 msec. The CSRF measurement was made under dark- and light- adapted conditions. A

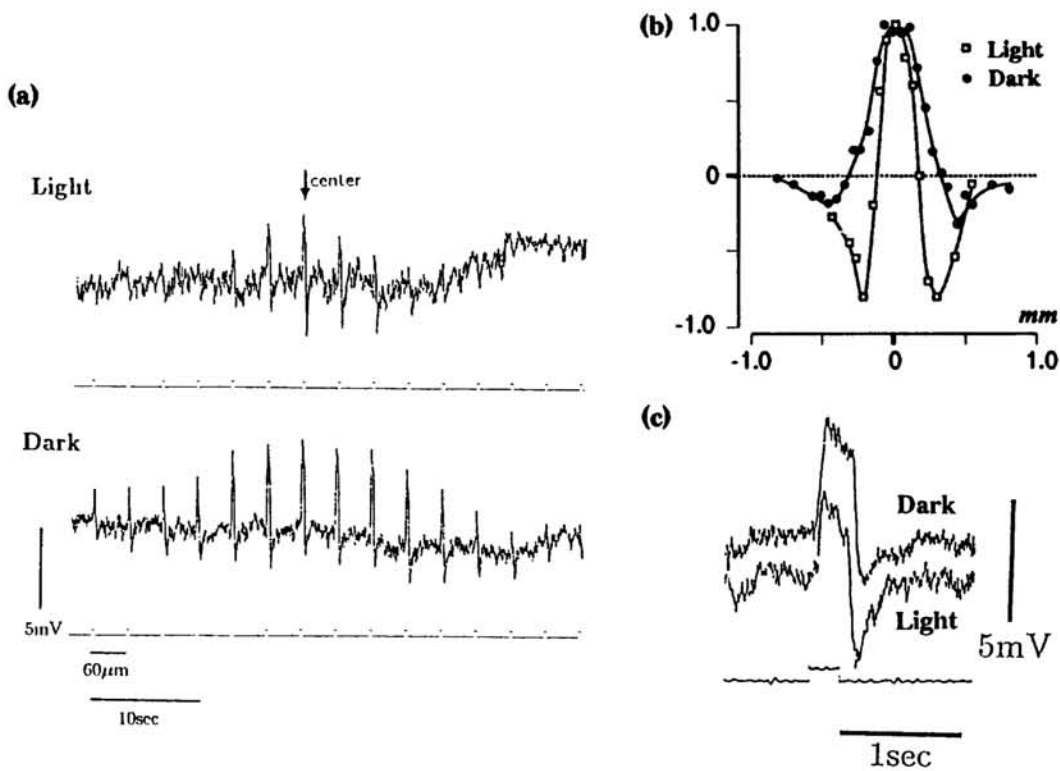

Figure 1: (a) Intracellular recordings from an ON-center bipolar cell of the carp retina with moving slit stimuli under light and dark adapted condition. (b) The receptive field profiles plotted from the recordings. (c) The response recorded when the slit was positioned at the receptive field center.

steady background light of 0.29 $\mu$W/cm$^2$ was provided for light adaptation.

## 2.2 RESULTS

Fig.1a shows a typical set of records obtained from a bipolar cell. The response to each flash of slit was biphasic (i.e., BPFR), consisting of a depolarization (ON) followed by a hyperpolarization(OFF). The ON response was the major component when the slit was positioned centrally on the receptive field, whereas the OFF response was dominant at peripheral locations and somewhat sluggish. The CSRF pattern was portrayed by plotting the response membrane potential measured at the time just prior to the cessation of each test flash. The result compiled from the data of Fig.1a is presented in Fig.1b, showing that the CSRF of the dark-adapted state was shallow and broad as opposed to the sharp profile produced during light adaptation. The records with the slit positioned at the receptive field center are enlarged in Fig.1c, indicating that the OFF part of the BPFR waveform was shallower and broader when the retina was dark adapted than when light adapted.

## 3   THE CASE OF ARTIFICIAL NEURAL NETWORKS

Visual pattern recognition and imagery data processing have been a traditional application area of ANNs. There are also ANNs that deal with time series signals. These both types of ANNs are considered here, and they are trained to detect and localize spatial or temporal changes of the input signal corrupted by noise.

## 3.1 PARADIGMS AND METHODS

The ANN models we used are illustrated in Figs.2. The model of Fig.2a deals with one-dimensional spatial signals. It consists of three layers (input, hidden, output), each having the same number of 12 or 20 neuronal units. The pattern given to the input layer represents the brightness distribution of light. The network was trained by means of the standard backpropagation algorithm, to detect and localize step-wise changes (edges) which were distributed on each training pattern in a random fashion with respect to the number, position and height. The mean level of the whole pattern was varied randomly as well. In addition, there was a background noise (not illustrated in Figs.2); independent noise signals of the same statistics were given to the all input units, and the maximum noise amplitude (NL: noise level) remained constant throughout each training session. The teacher signal was the "true" edge positions which were subject to obscuration due to the background noise; the learning was supervised such that each output unit would respond with 1 when a step-wise change not due to the background noise occurred at the corresponding position, and respond with −1 otherwise. The value of each synaptic weight parameter was given randomly at the outset and updated by using the backpropagation algorithm after presentation of each training pattern. The training session was terminated when the mean square error stopped decreasing.

To process time series inputs, the ANN model of Fig.2b was constructed with the backpropagation learning algorithm. This temporal model also has three layers, but the meaning of this is quite different from the spatial network model of Fig.2a. That is, whereas each unit of each layer in the spatial model is an anatomical entity, this is not the case with respect to the temporal model. Thus, each layer represents a single neuron so that there are actually only three neuronal elements, i.e., a receptor, an interneuron, and an output cell. And, the units in the same layer represent activity states of one neuron at different time slices; the rightmost unit for the present time, the next one for one time unit ago, and so on. As is apparent from Fig.2b, therefore, there is no convergence from the future (right) to the past (left). Each cell has memory of $T$-units time. Accordingly, the network requires $2T - 1$ units in the input layer, $T$ units in the hidden layer and 1 units in the output layer to calculate the output at present time. The input was a discrete time series in which step-wise changes took place randomly in a manner analogous to the spatial input of Fig.2a. As in the spatial case, there was a background noise

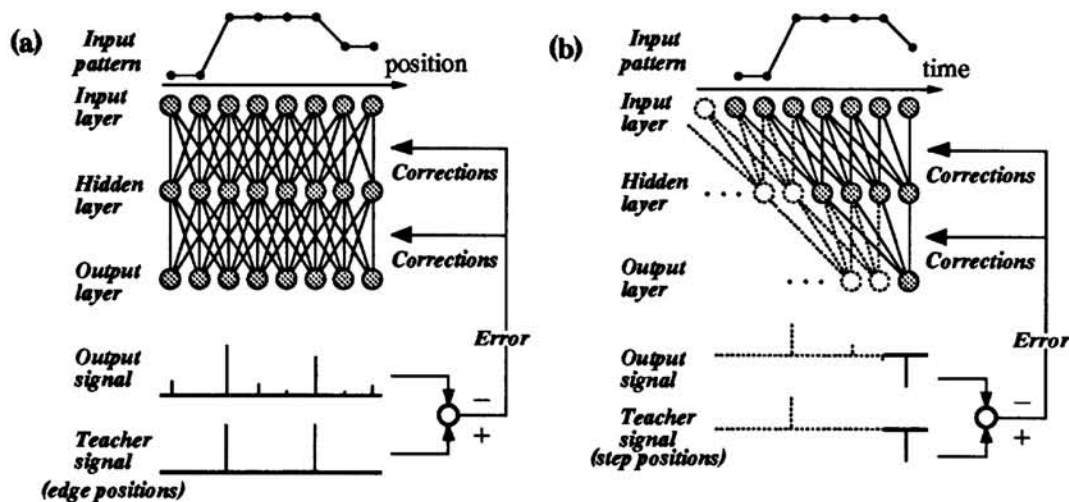

Figure 2: The neural network architectures. Spatial (a) and temporal model (b).

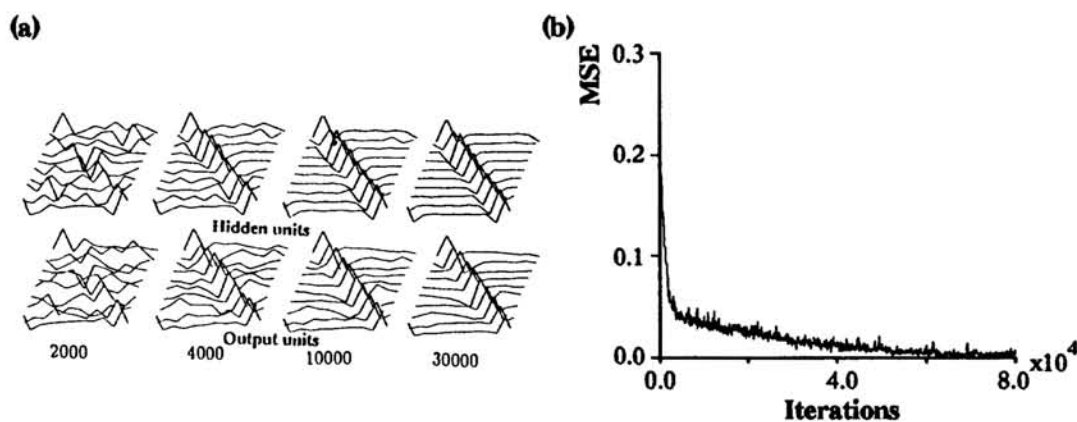

Figure 3: Development of receptive fields. Synaptic weights (a) and mean square error (b), both as a function of the number of iterations.

added to the input. The network was trained to respond with $+1/-1$ when the original input signal increased/decreased, and to respond with 0 otherwise.

## 3.2   RESULTS

### Spatial case: Emergence of CSRFs with SNR-dependent plasticity

As regards the edge detection learning by the ANN model of Fig.2a, the results without the background noise are described first (Furukawa & Yasui, 1990; Joshi & Lee, 1993). Fig.3a illustrates how the synaptic connections developed from the initial random state. If the final distribution of synaptic weight parameters is examined from input units to any hidden unit and also from hidden units to any output unit, then it can be seen in either case that the central and peripheral connections are opposite in the polarity of their weight parameters; the central group had either positive (ON-center) or negative (OFF-center) values, but the reversed profiles are

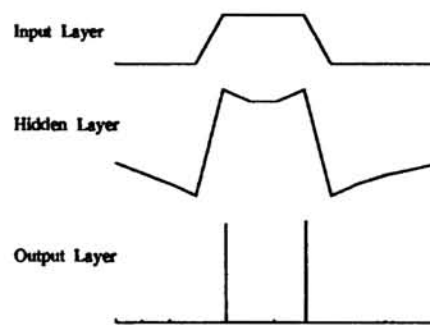

Figure 4: A Sample of activity pattern of each layer

shown in the drawing of Fig.3a for the OFF-center case. In any event, CSRFs were formed inside the network as a result of the edge detection learning. Fig.3b shows the performance improvement during a learning session. Fig.4 shows the activation pattern of each layer in response to a sample input, and edge enhancement like the Mach band effect can be observed in the hidden layer. Fig.5a presents sample input patterns corrupted by the background noise of various NL values, and Fig.5b shows how a hidden unit was connected to the input layer at the end of training. CSRFs were still formed when the environment suffered from the noise. However, the structure of the center-surround antagonism changed as a function of NL; the CSRFs became shallow and broad as NL increased, i.e., as the SNR decreased.

### Temporal case: Emergence of BPFRs with SNR-dependent plasticity

With reference to the learning paradigm of Fig.2b, Fig.5c reveals how a representative hidden unit made synaptic connections with the input units as a function of NL; the weight parameters are plotted against the elapsed time. Each trace would correspond to the response of the hidden unit to a flash of light, and it consists of

two phases of ON and OFF, i.e., BPFRs (biphasic flash responses) emerged in this ANN as a result of learning, and the biphasic time course changed depending on NL; the negative-going phase became shallower and longer with decrease of SNR.

## 4&emsp;DISCUSSION: Common Receptive Field Properties in Vertebrate, Invertebrate and Artificial Systems

A CSRF profile emerges after differentiating twice in space a small patch of light, and CSRF is a kind of point spreading function. Accordingly, the response to any input distribution can be obtained by convolving the input pattern with CSRF. The double differentiation of this spatial filtering acts to locate edge positions. On the other hand, the waveform of BPFR appears by differentiating once in time a short flash of light. Thus, the BPFR is an impulse response function with which to convolve the given input time series to obtain the response waveform. This is a derivative filtering, which subserves detection of temporal changes in the input visual signal. While both CSRF and BPFR occur in visual neurons of a wide variety of vertebrates and invertebrates, the first part of the present study shows that these spatial and temporal filtering functions can develop autonomously in our ANNs.

The neural system of visual signal processing encounters various kinds of noise. There are non-biological ones such as a background noise in the visual input itself and the photon noise which cannot be ignored when the light intensity is low. Endogenous sources of noise include spontaneous photoisomerization in photoreceptor cells, quantal transmitter release at synaptic sites, open/close activities of ion channels and so on. Generally speaking, therefore, since the surroundings are dim when the retina is dark adapted, SNR in the neuronal environment tends to be low during dark adaptation. According to the present experiment on the carp retina, the CSRF of bipolar cells widens in space and the BPFR is prolonged in time when the retina is dark adapted, that is, when SNR is presumably low. Interestingly, the same SNR-dependent properties have also been described in connection with the CSRF and BPFR of large monopolar cells in the fly compound eye. These spatial and temporal observations are both in accord with a notion that a method to remove noise is smoothing which requires averaging for a sufficiently long interval. In other words, when SNR is low, the signal averaging takes place over a large portion of the spatio-temporal domain comprised of CSRF and BPFR. Smoothing and differentiation are entirely opposite in the signal processing role. The SNR dependency of the CSRF and BPFR profiles can be viewed as a compromise between these two operations, for the need to detect signal changes in the presence of noise. These

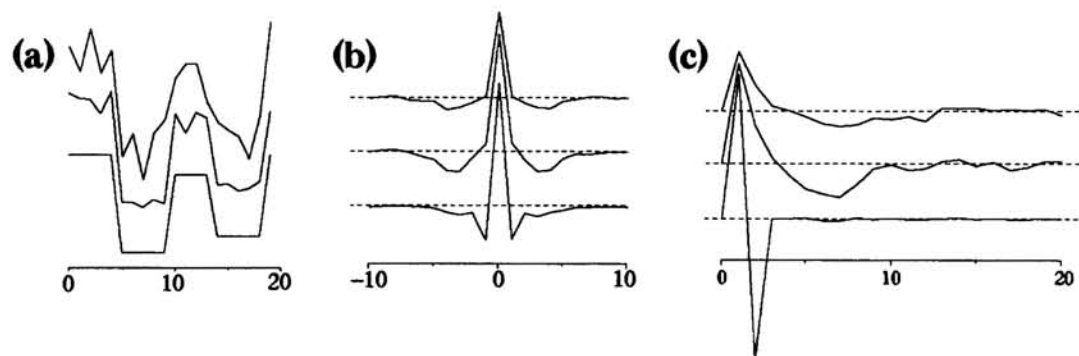

Figure 5: (a) A sample set of training patterns with different background noise levels (NLs). The NLs are 0.0, 0.4, 1.0 from bottom to top. The receptive field profiles (b) and flash responses (c) after training with each NL. The ordinate scale is linear but in arbitrary unit, with the zero level indicated by dotted lines.

points parallel the results of information-theoretic analysis by Atick and Redlich (1992) and by Laughlin (1982).

# 5  CONCLUDING REMARKS

We have learnt from this study that the same software is at work for the SNR-dependent control of the spatio-temporal visual receptive field in entirely different hardwares; namely, vertebrate, invertebrate and artificial neural systems. In other words, the plasticity scheme represents nature's optimum answer to the visual functional demand, not a result of compromise with other factors such as metabolism or morphology. Some mention needs to be made of the standard regularization theory. If the theory is applied to the edge detection problem, then one obtains the Laplacian-Gaussian filter which is a well-known CSRF example(Torre & Poggio, 1980). And, the shape of this spatial filter can be made wide or narrow by manipulating the value of a constant usually referred to as the regularization parameter. This parameter choice corresponds to the compromise that our ANN finds autonomously between smoothing and differentiation. The present type of research aided by trainable artificial neural networks seems to be a useful top-down approach to gain insight into the brain and neural mechanisms. Earlier, Lehky and Sejnowski (1988) were able to create neuron-like units similar to the complex cells of the visual cortex by using the backpropagation algorithm, however, the CSRF mechanism was given *a priori* to an early stage in their ANN processor. It should also be noted that Linsker (1986) succeeded in self-organization of CSRFs in an ANN model that operates under the learning law of Hebb. Perhaps, it remains to be examined whether the CSRFs formed in such an unsupervised learning paradigm might also possess an SNR-dependent plasticity similar to that described in this paper.

## References

Atick, J.J. & Redlich, A.N. (1992) What does the retina know about natural scenes? *Neural Computation, 4*, 196-210.

Dubs, A. (1982) The spatial integration of signals in the retina and lamina of the fly compound eye under different conditions of luminance. *J. Comp. Physiol A, 146*, 321–334.

Furukawa, T. & Yasui, S. (1990) Development of center-surround opponent receptive fields in a neural network through backpropagation training. *Proc. Int. Conf. Fuzzy Logic & Neural Networks* (Iizuka, Japan) 473–490.

Joshi, A. & Lee, C.H. (1993) Backpropagation learns Marr's operator *Biol. Cybern., 70*, 65–73.

Laughlin, S. B. (1982) Matching coding to scenes to enhance efficiency. In Braddick OJ, Sleigh AC(eds) *The physical and biological processing of images* (pp.42–52). Springer, Berlin, Heidelberg New York.

Lehky, S. R. & Sejnowski, T. J. (1988) Network model of shape-from shading: neural function arises from both receptive and projective fields. *Nature, 333*, 452–454.

Linsker, R. (1986) From basic network principles to neural architecture: Emergence of spatial-opponent cells. Proc. Natl. Acad. Sci. USA, 83, 7508–7512.

Stork, D. G. & Hall, J. (1989) Is backpropagation biologically plausible? *International Join Conf. Neural Networks, II (Washington DC)*, 241–246.

Torre, V. & Poggio, T. A. (1986) On edge detection. *IEEE Trans. Pattern Anal. Machine Intel., PAMI-8*, 147–163.
